# Attention Temperature Matters in ViT-Based Cross-Domain Few-Shot Learning

**Yixiong Zou**    **Ran Ma**    **Yuhua Li**    **Ruixuan Li**[*]

School of Computer Science and Technology, Huazhong University of Science and Technology
{yixiongz, ranma, idcliyuhua, rxli}@hust.edu.cn

## Abstract

Cross-domain few-shot learning (CDFSL) is proposed to transfer knowledge from large-scale source-domain datasets to downstream target-domain datasets with only a few training samples. However, Vision Transformer (ViT), as a strong backbone network to achieve many top performances, is still under-explored in the CDFSL task in its transferability against large domain gaps. In this paper, we find an interesting phenomenon of ViT in the CDFSL task: by simply multiplying a temperature (even as small as 0) to the attention in ViT blocks, the target-domain performance consistently increases, even though the attention map is downgraded to a uniform map. In this paper, we delve into this phenomenon for an interpretation. Through experiments, we interpret this phenomenon as a remedy for the ineffective target-domain attention caused by the query-key attention mechanism under large domain gaps. Based on it, we further propose a simple but effective method for the CDFSL task to boost ViT's transferability by resisting the learning of query-key parameters and encouraging that of non-query-key ones. Experiments on four CDFSL datasets validate the rationale of our interpretation and method, showing we can consistently outperform state-of-the-art methods. Our codes are available at https://github.com/Zoilsen/Attn_Temp_CDFSL.

## 1   Introduction

Deep networks have shown great power in learning from large-scale datasets [18, 11, 7]. However, collecting sufficient training data for every domain is always challenging, which gives rise to the Cross-Domain Few-Shot Learning (CDFSL) task. CDFSL requires a model to be firstly trained on a large-scale pretraining dataset (source domain, general dataset, e.g., ImageNet [6]), and then transferred to downstream datasets (target domain, expert-knowledge dataset, e.g., medical dataset [5, 44]) where only a few training samples are available. Typically, large domain gaps exist between the source and target dataset, making the transferring and downstream learning difficult [2, 12, 15].

Vision Transformer (ViT) [8], as a prevailing kind of deep network, has achieved top performances in many computer vision tasks [47, 9, 37]. However, only a few works [14, 10, 53] studied ViT's transferability against large domain gaps for the CDFSL task. In this paper, we find an intriguing phenomenon for the ViT-based CDFSL task: by multiplying a temperature parameter $\tau$ to the attention map of ViT blocks, the downstream target-domain few-shot performance consistently increases when $\tau < 1.0$ or even close to 0, albeit the attention map is downgraded to a uniform map (Fig. 1).

In this paper, we delve into this phenomenon for an interpretation. We find the query-key attention mechanism in the ViT network demonstrates high discriminability but low transferability, which makes the target-domain attention ineffective, and the temperature adjustment is interpreted as a remedy for the ineffective target-domain attention. Moreover, we find the non-query-key structures

---

[*]Corresponding author.

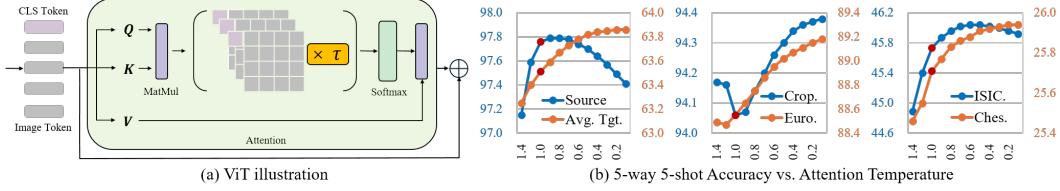

Figure 1: (a) Vision Transformer (ViT) is composed of multiple blocks of the MLP and attention networks. (b) We train a ViT model on *mini*ImageNet (source dataset), and conduct prototype-based classification on target datasets. By multiplying a temperature $\tau$ (even as small as 0) to the attention in ViT blocks (e.g., the last block) during the target-dataset classification, we find the accuracy consistently increases on most target datasets (Crop. Euro. ISIC. and Ches., even with a uniform attention map) while that of the source dataset drops. In this paper, we delve into this phenomenon for an interpretation, and propose a simple but effective method for the CDFSL task based on it.

in ViT show complementary characteristics against the query-key parts, i.e., higher transferability but lower discriminability. Based on these interpretations and findings, we propose a method for the CDFSL task to encourage the learning of non-query-key parameters and resist that of query-key ones, so as to improve the transferability of ViT against large domain gaps.

In summary, our contribution can be listed as

• To the best of our knowledge, we are the first to unveil the importance of the attention temperature in ViT-based CDFSL methods.

• Through experiments, we find the query-key attention mechanism shows limited transferability against large domain gaps, which causes ineffective target-domain attention and needs to be remedied by the temperature adjustment.

• Based on it, we propose a method for the CDFSL task to boost ViT's transferability by resisting the learning of query-key parameters and encouraging that of the non-query-key ones.

• Extensive experiments validate the rationale of our interpretation and method, and show that we can consistently outperform state-of-the-art works.

## 2 Delve into Attention in ViT-based Cross-Domain Few-Shot Learning

### 2.1 Preliminaries

Cross-Domain Few-Shot Learning (CDFSL) requires the model to first learn from a general dataset (i.e., source dataset) containing sufficient training samples, and then transfer to downstream target datasets where only scarce training data is available. We denote the source dataset as $D^S = \{x_i^S, y_i^S\}_{i=1}^N$ where $x_i^S$ and $y_i^S$ represent the $i$th training sample and its label, respectively. Similarly, the target dataset is termed as $D^T = \{x_i^T, y_i^T\}_{i=1}^{N'}$. During the learning and testing on $D^T$, for the fair comparison, current works [2, 12] adopt a $k$-way $n$-shot paradigm to sample from $D^T$ to construct small datasets (i.e., episodes) consisting of $k$ classes and $n$ training samples in each class. Based on episodes, the model learns from these $k \cdot n$ samples (a.k.a. support set, $\{x_{ij}, y_i\}_{i=1,j=1}^{k,n}$) and is evaluated on testing samples from these $k$ classes (a.k.a. query set, $\{x^q\}$).

Vision Transformer (ViT) has recently been popular in vision tasks, which divides images into tokens and captures the image patterns through self-attentions. As shown in Fig. 1, ViT is composed of stacked blocks containing an attention network and an MLP network, which can be represented as

$$f(x_i^S) = M(A(M(\cdots A(E(x_i^S))\cdots))), \tag{1}$$

where $M(\cdot)$ denotes the MLP network, $A(\cdot)$ denotes the self-attention network, and $E(\cdot)$ denotes the embedding layer. In this paper, we focus on the ViT-based CDFSL method to study its downstream generalization. Specifically, we follow [10] to initialize ViT on ImageNet [6] by DINO [1] (see appendix for other settings), and train ViT on $D^S$ by the cross-entropy loss with a fully-connected (FC) layer as

$$L = L_{cls}(\phi(f(x_i^S)), y_i^S), \tag{2}$$

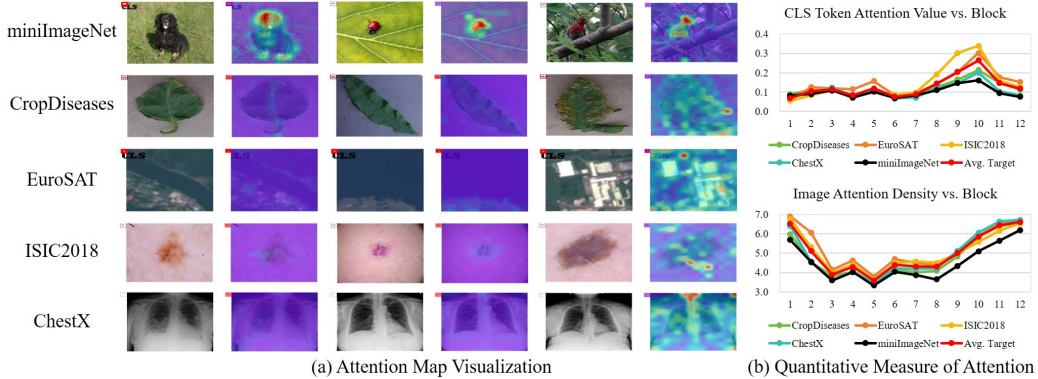

(a) Attention Map Visualization      (b) Quantitative Measure of Attention

Figure 2: (a) Visualization of ViT attention. Although ViT behaves well on the source domain (*mini*ImageNet), on target domains ViT tends to (1) wrongly focus on the CLS token while omitting the input images, and (2) dispersively focus on a large range of noisy image regions. (b) We quantitatively plot (top) the attention value of the CLS token and (bottom) the density of attention maps on image tokens *for all images in each domain*. The black curve of *mini*ImageNet is always below other curves of target domains, verifying the ineffectiveness of target-domain attention. Therefore, we interpret the temperature adjustment as a remedy for the ineffective target-domain attention.

where $\phi(\cdot)$ denotes the FC-based classifier and $f(\cdot)$ denotes ViT. Finally, we utilize the ProtoNet [33] with the distance function $d(\cdot, \cdot)$ for the target domain few-shot learning as

$$\hat{y}_q = \arg\min_i d(\frac{1}{n} \sum_j f(x_{ij}), f(x_q)). \tag{3}$$

In this section, we follow [12] to take *mini*ImageNet [40] as the source dataset, and take CropDiseases [30], EuroSAT [13], ISIC2018 [5] and ChestX [44] as target datasets.

## 2.2 Interpretation: Attention Temperature Remedies Target-Domain Attentions

### 2.2.1 Intuitive Observation of Ineffective Target-Domain Attentions

To study why the temperature-based attention adjustment improves on target datasets but harms the source dataset, we first visualize the attention map on each domain in Fig. 2a. As the backbone network utilizes the CLS token feature from the last ViT block as the final feature, we take the attention map w.r.t. the CLS token in the last ViT block for visualization, where a large attention value indicates the color of red and a small value refers to blue. We plot the CLS token's attention value in the left-top of the map. We can see the source-domain-trained ViT shows a good capability of discovering meaningful objects in the source dataset (*mini*ImageNet), but it always shows wrong attention on target datasets, which is represented in two aspects:

(1) It tends to excessively focus on the CLS token and ignores all image tokens, as the left-top patches are red but the image heatmap is blue for the first two columns. (2) For its focus on images, it tends to focus on a large range of noisy regions instead of meaningful objects (the third column).

Ideally, if there is no domain gap between the source and target domains, the target-domain attention should perform like the source-domain attention to focus on meaningful regions in the image. Therefore, these phenomena indicate that the attention network performs poorly on target domains.

### 2.2.2 Quantitative Verification of Target-Domain Attentions' Ineffectiveness

To verify this observation, we quantitatively measure (1) the attention value on the CLS token and (2) the sparsity of the attention on image tokens, on *all images from different datasets and blocks*. The attention value on the CLS token is measured as

$$V(A) = \frac{1}{b} \frac{1}{n_h} \frac{1}{n_t} \sum_{i,j,k} r(A_{i,j,k})_{[0]}, \tag{4}$$

where $A \in R^{b \times n_h \times n_t \times n_t}$ is the attention map, $b$ is the batch size, $n_h$ is the number of heads, $n_t$ is the number of tokens and $r(\cdot)$ denotes the $L_2$-normalization, Similarly, The sparsity of the image

Table 1: Ablation of the attention network from ViT's last block.

| Method | *mini*ImageNet | CropDiseases | EuroSAT | ISIC2018 | ChestX | Average |
|---|---|---|---|---|---|---|
| Input Tokens | 90.17 | 79.63 | 73.12 | 32.81 | 22.41 | 51.99 |
| Input Tokens + SA | 92.59 | 79.10 | 73.17 | 32.54 | 22.47 | 51.82 |
| Input Tokens + Identity SA | 87.45 | 77.97 | 69.89 | 32.15 | 22.52 | 50.63 |
| Input Tokens + Cosine SA | 88.80 | 79.98 | 74.35 | 32.65 | 22.57 | 52.39 |
| Input Tokens + Average SA | 89.53 | 80.73 | 74.59 | 32.04 | 22.64 | 52.50 |

Table 2: Domain similarity w.r.t. ablated attention modules.

| Method | *mini*ImageNet | CropDiseases | EuroSAT | ISIC2018 | ChestX | Average |
|---|---|---|---|---|---|---|
| Input Tokens | 1.0 | 0.4569 | 0.4381 | 0.3608 | 0.3900 | 0.4115 |
| Input Tokens + SA | 1.0 | 0.1853 | 0.1829 | 0.1344 | 0.1998 | 0.1756 |
| Input Tokens + Identity SA | 1.0 | 0.5857 | 0.5873 | 0.5376 | 0.4836 | 0.5486 |
| Input Tokens + Cosine SA | 1.0 | 0.2692 | 0.2252 | 0.1616 | 0.2295 | 0.2214 |
| Input Tokens + Average SA | 1.0 | 0.2235 | 0.2226 | 0.1580 | 0.2002 | 0.2011 |

attention is measured by the averaged $L_2$-normalized $L_1$ norm [54] of the attention map as

$$norm(A) = \frac{1}{b} \frac{1}{n_h} \frac{1}{n_t} \sum_{i,j,k} L_1(r(A_{i,j,k})_{[1:]}). \tag{5}$$

Results are plotted in Fig. 2b. As the smaller the $L_1$ norm is, the sparser (i.e., less dense) the attention would be, in Fig. 2b (bottom), we use density as the Y-axis to align with the $L_1$ norm value. We can see that curves of the source dataset (*mini*ImageNet) are always located under those of target datasets in Fig. 2b, indicating the model averagely pays more attention to the CLS token or noisy image regions on target datasets, quantitatively validating the ineffectiveness of target-domain attention.

Therefore, we interpret the temperature adjustment as a remedy for ineffective target-domain attention: If we apply a small temperature to the attention map, the attention map will be smoothed. For example, given the smallest temperature 0, the attention map will be downgraded to a uniform map. Such a smoothing operation would remedy the wrong attention on all tokens (i.e., CLS token and image tokens), because **uniform attention is at least better than wrong attention** (e.g., uniform attention allows the model to collect information from image tokens, but the wrong attention restricts the model to only collect information from the CLS token). These also interpret why the temperature remedy only improves on target datasets: the source dataset is effective in finding meaningful regions, indicating a good attention map that does not need to be remedied.

### 2.3 Why do attention networks get ineffective on target domains?

Then, we take the attention network in the last ViT block for ablation, and study the performance (5-way 1-shot accuracy, Tab. 1) and domain similarity (Tab. 2) induced by these modules. The domain similarity is measured by the CKA similarity [17, 23], where we extract features from images of different domains, and measure the CKA similarity between the source and each target dataset by aligning the channel dimension. Large domain similarities indicate less domain information contained. Since a trade-off exists between discriminability and transferability [19, 27, 46], we utilize the source-domain (*mini*ImageNet) accuracy in Tab. 1 to measure the discriminability and use the average domain similarity in Tab. 2 to measure the transferability, and the average target-domain accuracy in Tab. 1 is the result caused by the trade-off.

In Tab. 1, 2, Input Tokens denote features input into the attention network (i.e., the output of the 11th ViT block), and SA (self-attention) denotes the output of the attention network (i.e., the input of the MLP network in the 12th ViT block), which is the default ViT self-attention module. By appending SA to Input Tokens, in Tab. 1, the source-domain accuracy increases. Accordingly, in Tab. 2, the CKA similarity significantly drops, indicating SA is more on the side of discriminability than transferability, which therefore makes target-domain accuracies decrease.

Then, we study the token relation modeled by the self-attention mechanism. We replace the default query-key relation with three simpler relations: the identity relation (tokens are not merged), the cosine relation (tokens are merged by the cosine similarity with softmax), and the average relation (tokens are merged uniformly, equivalent to the temperature set to 0). Firstly, we can see the identity SA shows low source and target accuracy but high domain similarity, indicating the merge of tokens

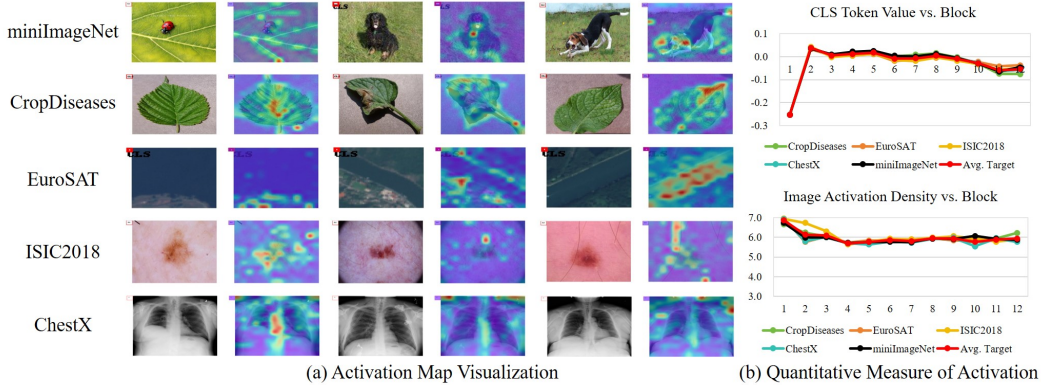

(a) Activation Map Visualization                    (b) Quantitative Measure of Activation

Figure 3: (a) We visualize the activation maps of Input Tokens in Tab. 1, and find that although the activation maps are not perfect, they show (1) no exceeding activations on the CLS token, and (2) rough object contours in input images. (b) We further find CLS token activation values as well as the map density on image tokens are non-distinguishable between source and target domains, indicating a better transferability of representations against large domain gaps. These indicate the Input Token in Tab. 1 tends to be more transferable than the query-key attention.

is crucial for discriminability. Therefore, if target-domain attention majorly focuses on the CLS token (Fig. 2), the merge of image tokens would be abandoned, leading to downgraded performance.

By substituting the Identity SA with the Cosine and the Average SA, the performance improves on all datasets, which is lower on the source dataset but much higher on target datasets than the default SA. Also, the domain similarity is higher than the default SA but is lower than the Identity SA, indicating:

(1) The modeling of token relations contains domain information and discriminability itself, therefore it contributes to the source domain and reduces the domain similarity. The increase in target-domain accuracies indicates the gain of discriminability is larger than the loss of transferability.

(2) The default query-key relation contains the most domain information and discriminability, which increases the source-domain performance the most and decreases the domain similarity the most. Therefore, the query-key relation even harms the target-domain performance.

As a result, we can conclude that ineffective target-domain attention is majorly caused by the self-attention mechanism in the attention network. Given the trade-off between discriminability and transferability, the modeling of token relations is not entirely harmful for target-domain generalization. However, the query-key attention mechanism tends to be much less transferable, and some related works [36] have partly shown the tendency of overfitting in this mechanism, therefore it may harm the generalization to target domains and lead to ineffective target-domain attention.

## 2.4   Handle the Ineffective Target-Domain Attention

Besides the query-key attention (SA) in Tab. 1 and 2, we can also find the rows of Input Tokens and Average SA demonstrate higher domain similarity and good target-domain performance, albeit the limited source-domain performance. This indicates these features, compared with the features of the query-key attention, tend to be on the transferability side in the trade-off between discriminability and transferability, since the query-key attention contains more learnable parameters.

To verify this hypothesis, we also visualize the activation map of the Input Tokens of Tab. 1 in Fig. 3a. We observe (1) no exceeding activation on the CLS token, and (2) the activation maps can indeed reflect rough object contours in images from different domains, although not perfect.

Moreover, we measure the image token density and the CLS token value on these features. Similar to $norm(A)$ and $V(A)$, these two criteria are written as $norm(M) = \frac{1}{b} \sum_i L_1(r(M_i)_{[1:]})$ and $V(M) = \frac{1}{b} \sum_i r(M_i)_{[0]}$, where $M \in R^{b \times n_t \times c}$ is the MLP outputs and $r(M) = \frac{1}{c} \sum M_{:,:,k}/||\frac{1}{c} \sum M_{:,:,k}|| \in R^{b \times n_t}$ is the $L_2$-normalized average activation of each token. The results are plotted in Fig. 3b, and we can see the source and target datasets are **non-distinguishable** on these two criteria, indicating better transferability and adequate discriminability.

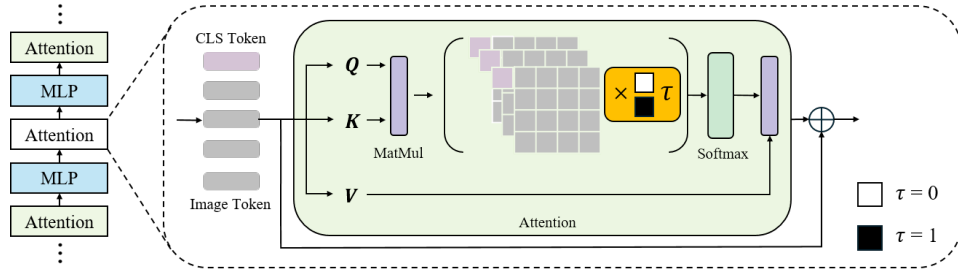

Figure 4: During the source-domain training, we propose to randomly abandon the attention network by multiplying a temperature of 0 to the attention in each block respectively, which resists the learning of the query-key attention parameters and enhances the non-query-key parts.

Therefore, we conclude that the query-key features tend to be discriminative but less transferable, while the non-query-key features tend to be transferable but less discriminative. This inspires us to encourage the learning of the non-query-key parameters in ViT (since their ability to be discriminative and overfitting is limited by parameters), and resist the learning of the query-key parts (as they are discriminative enough after the pretraining, so the source-domain training may not be needed).

## 2.5   Conclusion and Discussion

The self-attention mechanism is the core design of ViT, where the query-key attention mechanism is capable of fitting large-scale training data. However, such characteristic also limits the transferability against large domain gaps, leaving the target-domain attention ineffective. Therefore, a remedy of the attention map by the temperature adjustment is needed, as a uniform attention map is at least better than a wrong attention map. Compared with the query-key attention, the non-query-key components in ViT tend to be more transferable but less discriminative than the query-key components, which inspires us to improve the generalization of ViT by encouraging the learning of non-query-key parts and resisting the learning of query-key ones during the source-domain training.

## 3   Method

Based on the above analysis, we further propose a simple but effective method to boost ViT's transferability, which can be divided into a source-domain stage and a target-domain stage.

### 3.1   Source-Domain Attention Abandonment

Since the default query-key attention is verified to be vulnerable to domain gaps, we aim to enhance the non-query-key part of ViT in the source-domain training. We propose to stochastically abandon the query-key attention of all ViT blocks during the source-domain training, by multiplying a temperature of 0 (Fig. 4), so that the attention will be randomly downgraded to a uniform map. This is written as

$$A(x) = softmax(A'(x) \cdot \tau^S(p)), \tag{6}$$

where $A'(x) \in R^{b \times n_h \times n_t \times n_t}$ denotes the un-normalized attention map in Eq. 1, $\tau^S(p) \in \{0, 1\}$ is a scalar sampled from the binary distribution, and $p$ (typically set to 0.8) is the probability of being 0.

This operation will resist the learning of the query-key attention parameters (i.e., Q and K in Fig. 4), as query-key attention will be applied in the forward pass with a probability of $1 - p$. This operation will also encourage the learning of the non-query-key attention parameters, as the average attention is applied with a probability of $p$. Moreover, it will also resist the exceeding attention on the CLS token, as the CLS token's attention value in the uniform map would be $1/n_t$ which is very small. Note that it is different from the Dropout [34] operation in (1) Dropout is carried out element-wisely in the attention map, but we directly shift the whole attention map to a uniform map; (2) Dropout is conducted after the softmax function, but ours is before it.

### 3.2   Target-Domain Attention Adjustment

During the finetuning and evaluation of target datasets, the default query-key attention is still applied in the network, but we follow section 2 to set a pre-defined hyper-parameter $\tau^T$ for ViT blocks to

alleviate the influence of ineffective attention maps. This operation is written as

$$A(x) = softmax(A'(x) \cdot \tau^T). \tag{7}$$

Finally, we follow current works [12, 4] to conduct the prototype-based classification on the target-domain episodes as Eq. 3, or finetune our model on the support set of these episodes for the classifier-based classification.

## 4 Experiments

Table 3: Comparison with state-of-the-art works by the 5-way 1-shot classification.

| Method | backbone | FT | Mark | Crop. | Euro. | ISIC. | Ches. | Ave. |
|---|---|---|---|---|---|---|---|---|
| GNN+AFA [15] | ResNet10 | × | ECCV-22 | 67.61 | 63.12 | 33.21 | 22.92 | 46.97 |
| LDP-net [50] | ResNet10 | × | CVPR-23 | 69.64 | 65.11 | 33.97 | 23.01 | 47.18 |
| FLoR [53] | ResNet10 | × | CVPR-24 | 73.64 | 62.90 | **38.11** | 23.11 | 49.69 |
| SDT [29] | ResNet10 | × | NN-24 | 73.92 | 65.87 | 36.45 | **23.22** | 49.97 |
| MEM-FS [42] | ViT-S | × | TIP-23 | 81.11 | 68.11 | 32.97 | 22.76 | 51.24 |
| StyleAdv [10] | ViT-S | × | CVPR-23 | 81.22 | 72.15 | 33.05 | 22.92 | 52.34 |
| SDT [29] | ViT-S | × | NN-24 | 81.03 | 72.71 | 33.40 | 22.79 | 52.48 |
| FLoR [53] | ViT-S | × | CVPR-24 | 81.81 | 72.39 | 34.20 | 22.78 | 52.80 |
| **AttnTemp** | ViT-S | × | **Ours** | **84.02** | **74.35** | 34.92 | 23.19 | **54.12** |
| FLoR [53] | ResNet10 | ✓ | CVPR-24 | 84.04 | 69.13 | **38.81** | 23.12 | 53.78 |
| PMF [14] | ViT-S | ✓ | CVPR-22 | 80.79 | 70.74 | 30.36 | 21.73 | 50.91 |
| FLoR [53] | ViT-S | ✓ | CVPR-24 | 83.55 | 73.09 | 35.49 | 23.26 | 53.85 |
| StyleAdv [10] | ViT-S | ✓ | CVPR-23 | 84.11 | 74.93 | 33.99 | 22.92 | 53.99 |
| **AttnTemp** | ViT-S | ✓ | **Ours** | **84.78** | **75.09** | 38.05 | **23.63** | **55.39** |
| LDP-net* [50] | ResNet10 | ✓ | CVPR-23 | 81.24 | 73.25 | 33.44 | 22.21 | 52.54 |
| RDC* [22] | ResNet10 | ✓ | CVPR-22 | 85.79 | 70.51 | 36.28 | 22.32 | 53.73 |
| FLoR* [53] | ResNet10 | ✓ | CVPR-24 | 86.30 | 71.38 | **41.67** | 23.12 | 55.62 |
| MEM-FS+RDA* [42] | ViT-S | ✓ | TIP-23 | 83.74 | 75.91 | 37.07 | 23.85 | 55.14 |
| **AttnTemp*** | ViT-S | ✓ | **Ours** | **87.58** | **77.40** | 40.13 | **23.96** | **57.23** |

### 4.1 Datasets

Following current works [2, 12], we utilize the *mini*ImageNet dataset [40] as the source dataset, and utilize 4 cross-domain datasets as the target datasets, including CropDiseases [30], EuroSAT [13], ISIC2018 [5] and ChestX [44] for few-shot training and evaluation, using the $k$-way $n$-shot classification as stated in section 2.1. *mini*Imagenet [40] is a subset of the large-scale ImageNet [6] dataset with 100 categories and 60,000 images, where 64 categories are utilized for training. CropDiseases [30] is a dataset for agricultural disease recognition, consisting of 38 categories and 43,456 images. EuroSAT [13] is a satellite image dataset for land classification, comprising 10 classes and 27,000 images. ISIC2018 [5] is for skin lesion recognition, comprising 7 categories with 10,015 images. ChestX [44] contains chest X-ray images, with 7 categories and 25,847 images.

### 4.2 Implementation Details

We follow StyleAdv [10] to take ViT-S as the backbone network and take the DINO [1] pretraining on ImageNet as the initialization (other pretraining can be found in the appendix). During the source-domain training, we apply all ViT blocks to the Attention Abandonment. We use the Adam [16] optimizer with a learning rate of 0.001 for the classifier and $10^{-6}$ for the backbone network. During the target-domain few-shot evaluation, we set the temperature for the first two blocks as 0.3, and set the attention of the CLS token to 0 for blocks whose ID is greater than 4.

### 4.3 Comparison with State-of-the-Art Works

Tab. 3 and 4 report our results compared with state-of-the-art works for both 1-shot and 5-shot settings. We separately compare works with and without finetuning (FT) for fairness. The asterisk (*) denotes a transductive setting. PMF [14], StyleAdv [10], MEM-FS [42] and FLoR [53] are introduced for comparison. For all results, our work achieves the top average performance in all

Table 4: Comparison with state-of-the-art works by the 5-way 5-shot classification.

| Methods | backbone | FT | Mark | Crop. | Euro. | ISIC. | Ches. | Ave. |
|---|---|---|---|---|---|---|---|---|
| LDP-net [50] | ResNet10 | × | CVPR-23 | 89.40 | 82.01 | 48.06 | 26.67 | 61.29 |
| GNN+AFA [15] | ResNet10 | × | ECCV-22 | 88.06 | 85.58 | 46.01 | 25.02 | 61.67 |
| SDT [29] | ResNet10 | × | NN-24 | 90.27 | 82.02 | 48.66 | 27.20 | 62.04 |
| FLoR [53] | ResNet10 | × | CVPR-24 | 91.25 | 80.87 | 51.44 | 26.70 | 62.32 |
| MEM-FS [42] | ViT-S | × | TIP-23 | 93.74 | 86.49 | 47.38 | 26.67 | 63.57 |
| StyleAdv [10] | ViT-S | × | CVPR-23 | 94.85 | 88.57 | 47.73 | 26.97 | 64.53 |
| MICM [49] | ViT-S | × | MM-24 | 94.61 | 90.08 | 46.85 | 27.11 | 64.66 |
| SDT [29] | ViT-S | × | NN-24 | 95.00 | 89.60 | 47.64 | 26.72 | 64.75 |
| FLoR [53] | ViT-S | × | CVPR-24 | 95.28 | **90.41** | 49.52 | 26.71 | 65.48 |
| **AttnTemp** | ViT-S | × | **Ours** | **95.53** | 90.13 | **53.09** | **27.72** | **66.62** |
| FLoR [53] | ResNet10 | ✓ | CVPR-24 | 92.33 | 83.06 | **56.74** | 26.77 | 64.73 |
| PMF [14] | ViT-S | ✓ | CVPR-22 | 92.96 | 85.98 | 50.12 | 27.27 | 64.08 |
| StyleAdv [10] | ViT-S | ✓ | CVPR-23 | 95.99 | 90.12 | 51.23 | 26.97 | 66.08 |
| FLoR [53] | ViT-S | ✓ | CVPR-24 | 96.47 | 90.75 | 53.06 | 27.02 | 66.83 |
| **AttnTemp** | ViT-S | ✓ | **Ours** | **96.66** | **90.82** | 54.91 | **28.03** | **67.61** |
| LDP-net* [50] | ResNet10 | ✓ | CVPR-23 | 91.89 | 84.05 | 48.44 | 26.88 | 62.82 |
| RDC* [22] | ResNet10 | ✓ | CVPR-22 | 93.30 | 84.29 | 49.91 | 25.07 | 63.14 |
| FLoR* [53] | ResNet10 | ✓ | CVPR-24 | 93.60 | 83.76 | **57.54** | 26.89 | 65.45 |
| MEM-FS+RDA* [42] | ViT-S | ✓ | TIP-23 | 95.04 | 88.77 | 51.02 | 27.98 | 65.70 |
| **AttnTemp*** | ViT-S | ✓ | **Ours** | **96.74** | **91.34** | 55.22 | **28.41** | **67.93** |

Table 5: Ablation study by the 5-way 5-shot accuracy.

| Adjustment | Abandonment | CropDisease | EuroSAT | ISIC2018 | ChestX | Ave. |
|---|---|---|---|---|---|---|
| | | $94.24_{\pm0.27}$ | $88.62_{\pm0.22}$ | $45.72_{\pm0.33}$ | $25.66_{\pm0.17}$ | $63.53_{\pm0.13}$ |
| ✓ | | $94.48_{\pm0.31}$ | $88.73_{\pm0.25}$ | $49.12_{\pm0.28}$ | $25.81_{\pm0.21}$ | $64.53_{\pm0.20}$ |
| | ✓ | $95.40_{\pm0.33}$ | $89.08_{\pm0.29}$ | $52.01_{\pm0.31}$ | $27.49_{\pm0.19}$ | $66.00_{\pm0.19}$ |
| ✓ | ✓ | $\mathbf{95.53}_{\pm0.22}$ | $\mathbf{90.13}_{\pm0.33}$ | $\mathbf{53.09}_{\pm0.18}$ | $\mathbf{27.72}_{\pm0.19}$ | $\mathbf{66.62}_{\pm0.19}$ |
| Learnable Temp. [52] | | $94.26_{\pm0.08}$ | $88.91_{\pm0.09}$ | $46.45_{\pm0.11}$ | $26.44_{\pm0.08}$ | $64.02_{\pm0.07}$ |
| Dropout [34] | | $94.44_{\pm0.17}$ | $89.62_{\pm0.22}$ | $46.03_{\pm0.16}$ | $26.34_{\pm0.18}$ | $64.11_{\pm0.17}$ |
| Masking Diagonal Atten. [20] | | $94.83_{\pm0.08}$ | $88.95_{\pm0.09}$ | $46.98_{\pm0.12}$ | $26.74_{\pm0.09}$ | $64.38_{\pm0.17}$ |
| Fix Attention | | $94.67_{\pm0.22}$ | $89.66_{\pm0.20}$ | $46.22_{\pm0.19}$ | $26.31_{\pm0.20}$ | $64.08_{\pm0.19}$ |
| Totally abandon attention | | $95.07_{\pm0.18}$ | $88.90_{\pm0.23}$ | $49.23_{\pm0.19}$ | $27.44_{\pm0.14}$ | $65.16_{\pm0.16}$ |

settings and achieves the highest performance in almost all datasets. This demonstrates that our method effectively reduces the domain gap and enhances model transferability.

## 4.4 Ablation Study

We first ablate the Attention Adjustment and Abandonment. In Tab. 5 we can see both modules contribute to the performance, especially for the Attention Abandonment module which yields a significant 2.47% average improvement. By adding the Adjustment module, the performance also increases but is smaller than that applied to the baseline model. This is because by applying Attention Abandonment, the attention is improved so that the remedy by temperature is not in great need.

Then, we ablate our designs and compare them with other attention-temperature-based works.

(1) **Learnable or stochastic temperature?** [52] applied a small network to dynamically learn temperatures, which shows only trivial improvements, showing the importance of stochastic temperature.

(2) **Global or element-wise temperature?** Dropout [34] element-wisely abandons the attention after softmax and LSA [20] masks the diagonal attention map, which shows only marginal improvements on CDFSL, indicating the importance of a global temperature for each attention map.

(3) **Train attention or not?** Since abandoning the attention network can reduce its training and alleviate overfitting, we directly fix the attention network for verification and see only slight performance improvements, as the training of non-query-key parameters is limited by ineffective attention.

(4) **Maintain attention or not?** Due to the ineffectiveness of attention, we try to directly abandon all attention by setting a temperature of 0 for all blocks. We can see the performance is improved by

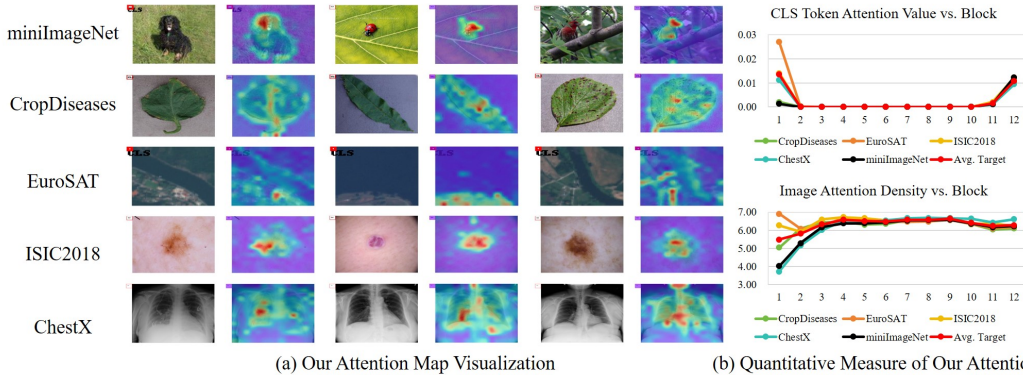

(a) Our Attention Map Visualization  (b) Quantitative Measure of Our Attention

Figure 5: (a) Visualization of attention maps of our model. We can see images activated only on the CLS token in Fig. 2a are now correctly activated, and the model can focus on the meaningful and concentrated regions, verifying the improved attention networks. (b) By evaluating the image-token-attention density and the CLS-token-attention value of our model, we can see these criteria are non-distinguishable between the source and target domains (compared with Fig. 2b), indicating attention networks' transferability against domain gaps is improved.

Table 6: Verification of improved self-attention w.r.t. domain similarity and target-domain accuracy.

| Metric. | 1 | 2 | 3 | 4 | 5 | 6 | 7 | 8 | 9 | 10 | 11 | 12 |
|---|---|---|---|---|---|---|---|---|---|---|---|---|
| BL CKA | 0.9805 | 0.9500 | 0.9667 | 0.9654 | 0.9455 | 0.9146 | 0.8940 | 0.8406 | 0.7446 | 0.6337 | 0.2063 | 0.1756 |
| Ours CKA | 0.9857 | 0.9590 | 0.9659 | 0.9704 | 0.9547 | 0.9347 | 0.9148 | 0.8763 | 0.7903 | 0.6655 | 0.2955 | 0.1886 |
| BL Acc. | 34.67 | 39.88 | 42.19 | 44.73 | 47.20 | 48.93 | 50.05 | 50.98 | 52.60 | 53.02 | 52.03 | 51.82 |
| Ours Acc. | 34.91 | 40.47 | 43.01 | 45.19 | 47.28 | 48.93 | 50.40 | 51.47 | 53.26 | 54.34 | 53.98 | 53.70 |

more than 1%, indicating relying on the non-query-key outputs can indeed help the cross-domain transferring. However, the accuracy is still lower than Attention Abandonment, showing that the query-key attention network still contains useful information for classification.

## 4.5 Verification of Improved Attention

### 4.5.1 Qualitative Study

We visualize attention maps of our model on both the source and target domain in Fig. 5a. In contrast to Fig. 2a, where attention primarily focuses on the CLS token in the target domain, our model can correctly activate meaningful and discriminative tokens. Furthermore, compared to the dispersed attention observed in Fig. 2a, our model focuses on more concentrated regions within the image, indicating that our model effectively transfers the attention network from the source to target domains.

### 4.5.2 Quantitative Study

We compare the image token attention density and the CLS token attention value mentioned in Sec. 2.2. As depicted in Fig. 5b, these criteria are mostly consistent between the source and target domain, in contrast to the disparities in Fig.2b, suggesting improved transferability of attentions.

Moreover, in Tab. 6, we report the domain similarity and target-domain accuracy of the features output by the attention network from each ViT block. Following Tab. 2, we measure the domain similarity by the CKA similarity. From Tab. 6, we can see our model improves the domain similarity of each self-attention's output, indicating improved transferability of attention networks. Consistently, the target-domain accuracy is also improved by our method.

Finally, following Tab. 1, we compare the domain similarity and performance between the default query-key attention and other attention choices in Tab. 6ab. We can see the Input feature of the attention network's CKA is decreased, as we encourage the learning of non-query-key parameters in ViT. However, such a decrease in CKA is much smaller than the decrease brought by the query-key attention. In contrast, the CKA of the attention network's output consistently increases, indicating better transferability against domains. This can also be verified in Tab. 6b that the performance difference between different attention choices is narrowed.

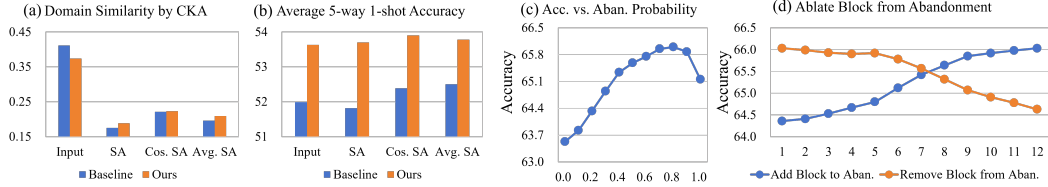

Figure 6: (a) Domain similarity of self-attention outputs consistently increases. (b) The difference in accuracy between different attention choices is narrowed. (c) A high probability of abandoning query-key attention can help the transferability. (d) All blocks need to be put into Attention Abandonment.

### 4.5.3 Sensitivity Study of Hyper-parameters

We plot the average target-domain accuracy vs. abandonment probability $p$ in Fig. 6c. When $p = 0$, it is downgraded to the baseline method. When $p = 1$, it means all attentions are abandoned. A high probability of abandoning can help the cross-domain transferring, but we cannot simply abandon all attention, indicating useful information in the attention network. Moreover, we ablate ViT blocks added to the Abandonment method in Fig. 6d. By gradually adding or removing abandoned blocks, the accuracy of the target domain increases or decreases accordingly. This indicates each block positively contributes to the model, highlighting the necessity of all blocks for Attention Abandonment.

## 5 Related Work

**Cross-Domain Few-Shot Learning** (CDFSL) aims to acquire knowledge from the target domain with limited training samples [2, 12]. The domain gaps between source and target domains make it challenging. Current works can be categorized into two groups: meta-learning based [2, 38, 43, 15], which simulates the data structure for the target-domain learning, and transferring-based [12, 32, 28, 25], involving training a model with strong generalization. However, they are mainly limited to the CNN structure, and while recent works [10, 42, 14, 45] have begun to utilize the transformer architecture for CDFSL tasks, they have not fully leveraged the potential of the ViT architecture.

**Domain Generalization** (DG) aims to generalize models from seen to unseen domains [41, 24], aligning with the objective of CDFSL. Recently, transformer-based approaches have been studied [48, 21, 31, 35]. [36] discovered that self-attention is not indispensable. [46] discovered that self-attention is not adept at distinguishing the transferability and discriminability of features across different domains. Different from them, we explore the influence of temperature on attention transferability against large domain gaps, without introducing any additional modules.

**Attention Temperature** can adjust the smoothness of the Softmax output distribution in the attention network. [39] proposes using a constant temperature to scale the dot product to alleviate a small extreme gradient. Recently, some methods have emerged to dynamically adjust the temperature while training deep learning models [26, 20, 3]. [20] proposes to apply a learnable temperature to attention scores to address overly smooth distributions. [3] reduces attention noise by suppressing accumulated trivial attention weights. In contrast, our study is the first, to our knowledge, to delve into the impact of attention temperature on cross-domain transferability.

## 6 Conclusion

In this paper, we find a phenomenon for the temperature-based attention adjustment in the ViT-based CDFSL task and delve into it for an interpretation. We interpret it as a remedy for the ineffective target-domain attention caused by the default query-key attention mechanism. Based on it, we further propose a method for CDFSL. Experiments validate our rationale and effectiveness.

## Acknowledgments

This work is supported by the National Natural Science Foundation of China under grants 62206102, 62436003, 62376103 and 62302184; the Science and Technology Support Program of Hubei Province under grant 2022BAA046; Hubei Science and Technology Talent Service Project under grant 2024DJC078; and Ant Group through CCF-Ant Research Fund.

# References

[1] Mathilde Caron, Hugo Touvron, Ishan Misra, Hervé Jégou, Julien Mairal, Piotr Bojanowski, and Armand Joulin. Emerging properties in self-supervised vision transformers. In *Proceedings of the International Conference on Computer Vision (ICCV)*, 2021.

[2] Wei-Yu Chen, Yen-Cheng Liu, Zsolt Kira, Yu-Chiang Frank Wang, and Jia-Bin Huang. A closer look at few-shot classification. In *Proceedings of the International Conference on Learning Representations*, 2018.

[3] Xiangyu Chen, Qinghao Hu, Kaidong Li, Cuncong Zhong, and Guanghui Wang. Accumulated trivial attention matters in vision transformers on small datasets. In *Proceedings of the IEEE/CVF Winter Conference on Applications of Computer Vision (WACV)*, pages 3984–3992, January 2023.

[4] Yinbo Chen, Zhuang Liu, Huijuan Xu, Trevor Darrell, and Xiaolong Wang. Meta-baseline: Exploring simple meta-learning for few-shot learning. In *Proceedings of the IEEE/CVF International Conference on Computer Vision (ICCV)*, pages 9062–9071, October 2021.

[5] Noel Codella, Veronica Rotemberg, Philipp Tschandl, M Emre Celebi, Stephen Dusza, David Gutman, Brian Helba, Aadi Kalloo, Konstantinos Liopyris, Michael Marchetti, et al. Skin lesion analysis toward melanoma detection 2018: A challenge hosted by the international skin imaging collaboration (isic). *arXiv preprint arXiv:1902.03368*, 2019.

[6] Jia Deng, Wei Dong, Richard Socher, Li-Jia Li, Kai Li, and Li Fei-Fei. Imagenet: A large-scale hierarchical image database. In *Proceedings of the IEEE/CVF Conference on Computer Vision and Pattern Recognition*, pages 248–255. Ieee, 2009.

[7] Jacob Devlin, Ming-Wei Chang, Kenton Lee, and Kristina Toutanova. Bert: Pre-training of deep bidirectional transformers for language understanding. In *Proceedings of the Conference of the North American Chapter of the Association for Computational Linguistics: Human Language Technologies*, pages 4171–4186, 2019.

[8] Alexey Dosovitskiy, Lucas Beyer, Alexander Kolesnikov, Dirk Weissenborn, Xiaohua Zhai, Thomas Unterthiner, Mostafa Dehghani, Matthias Minderer, Georg Heigold, Sylvain Gelly, Jakob Uszkoreit, and Neil Houlsby. An image is worth 16x16 words: Transformers for image recognition at scale, 2021.

[9] Haoqi Fan, Bo Xiong, Karttikeya Mangalam, Yanghao Li, Zhicheng Yan, Jitendra Malik, and Christoph Feichtenhofer. Multiscale vision transformers. In *Proceedings of the IEEE/CVF International Conference on Computer Vision (ICCV)*, pages 6824–6835, October 2021.

[10] Yuqian Fu, Yu Xie, Yanwei Fu, and Yu-Gang Jiang. Styleadv: Meta style adversarial training for cross-domain few-shot learning. In *Proceedings of the IEEE/CVF Conference on Computer Vision and Pattern Recognition (CVPR)*, pages 24575–24584, June 2023.

[11] Ian Goodfellow, Yoshua Bengio, and Aaron Courville. *Deep learning*. MIT Press, 2016.

[12] Yunhui Guo, Noel C Codella, Leonid Karlinsky, James V Codella, John R Smith, Kate Saenko, Tajana Rosing, and Rogerio Feris. A broader study of cross-domain few-shot learning. In *Proceedings of the IEEE/CVF European Conference on Computer Vision*, pages 124–141. Springer, 2020.

[13] Patrick Helber, Benjamin Bischke, Andreas Dengel, and Damian Borth. Eurosat: A novel dataset and deep learning benchmark for land use and land cover classification. *IEEE Journal of Selected Topics in Applied Earth Observations and Remote Sensing*, 12(7):2217–2226, 2019.

[14] Shell Xu Hu, Da Li, Jan Stühmer, Minyoung Kim, and Timothy M. Hospedales. Pushing the limits of simple pipelines for few-shot learning: External data and fine-tuning make a difference. In *Proceedings of the IEEE/CVF Conference on Computer Vision and Pattern Recognition (CVPR)*, pages 9068–9077, June 2022.

[15] Yanxu Hu and Andy J. Ma. Adversarial feature augmentation for cross-domain few-shot classification. In Shai Avidan, Gabriel Brostow, Moustapha Cissé, Giovanni Maria Farinella, and Tal Hassner, editors, *Computer Vision – ECCV 2022*, pages 20–37, Cham, 2022. Springer Nature Switzerland.

[16] Diederik P Kingma. Adam: A method for stochastic optimization. *arXiv preprint arXiv:1412.6980*, 2014.

[17] Simon Kornblith, Mohammad Norouzi, Honglak Lee, and Geoffrey Hinton. Similarity of neural network representations revisited. In *International Conference on Machine Learning*, pages 3519–3529. PMLR, 2019.

[18] Alex Krizhevsky, Ilya Sutskever, and Geoffrey E Hinton. Imagenet classification with deep convolutional neural networks. *Communications of the ACM*, 60(6):84–90, 2017.

[19] Jogendra Nath Kundu, Akshay R Kulkarni, Suvaansh Bhambri, Deepesh Mehta, Shreyas Anand Kulkarni, Varun Jampani, and Venkatesh Babu Radhakrishnan. Balancing discriminability and transferability for source-free domain adaptation. In Kamalika Chaudhuri, Stefanie Jegelka, Le Song, Csaba Szepesvari, Gang Niu, and Sivan Sabato, editors, *Proceedings of the 39th International Conference on Machine Learning*, volume 162 of *Proceedings of Machine Learning Research*, pages 11710–11728. PMLR, 17–23 Jul 2022.

[20] Seung Hoon Lee, Seunghyun Lee, and Byung Cheol Song. Vision transformer for small-size datasets. *arXiv preprint arXiv:2112.13492*, 2021.

[21] Bo Li, Yifei Shen, Jingkang Yang, Yezhen Wang, Jiawei Ren, Tong Che, Jun Zhang, and Ziwei Liu. Sparse mixture-of-experts are domain generalizable learners. In *The Eleventh International Conference on Learning Representations*, 2023.

[22] Pan Li, Shaogang Gong, Chengjie Wang, and Yanwei Fu. Ranking distance calibration for cross-domain few-shot learning. In *Proceedings of the IEEE/CVF Conference on Computer Vision and Pattern Recognition (CVPR)*, pages 9099–9108, June 2022.

[23] Wei-Hong Li, Xialei Liu, and Hakan Bilen. Universal representation learning from multiple domains for few-shot classification. In *Proceedings of the IEEE/CVF International Conference on Computer Vision (ICCV)*, pages 9526–9535, October 2021.

[24] Yiying Li, Yongxin Yang, Wei Zhou, and Timothy Hospedales. Feature-critic networks for heterogeneous domain generalization. In Kamalika Chaudhuri and Ruslan Salakhutdinov, editors, *Proceedings of the 36th International Conference on Machine Learning*, volume 97 of *Proceedings of Machine Learning Research*, pages 3915–3924. PMLR, 09–15 Jun 2019.

[25] Hanwen Liang, Qiong Zhang, Peng Dai, and Juwei Lu. Boosting the generalization capability in cross-domain few-shot learning via noise-enhanced supervised autoencoder. In *Proceedings of the IEEE/CVF International Conference on Computer Vision (ICCV)*, pages 9424–9434, October 2021.

[26] Junyang Lin, Xu Sun, Xuancheng Ren, Muyu Li, and Qi Su. Learning when to concentrate or divert attention: Self-adaptive attention temperature for neural machine translation. In Ellen Riloff, David Chiang, Julia Hockenmaier, and Jun'ichi Tsujii, editors, *Proceedings of the 2018 Conference on Empirical Methods in Natural Language Processing*, pages 2985–2990, Brussels, Belgium, October-November 2018. Association for Computational Linguistics.

[27] Bin Liu, Yue Cao, Yutong Lin, Qi Li, Zheng Zhang, Mingsheng Long, and Han Hu. Negative margin matters: Understanding margin in few-shot classification. In Andrea Vedaldi, Horst Bischof, Thomas Brox, and Jan-Michael Frahm, editors, *Computer Vision – ECCV 2020*, pages 438–455, Cham, 2020. Springer International Publishing.

[28] Bingyu Liu, Zhen Zhao, Zhenpeng Li, Jianan Jiang, Yuhong Guo, and Jieping Ye. Feature transformation ensemble model with batch spectral regularization for cross-domain few-shot classification. *arXiv preprint arXiv:2005.08463*, 2020.

[29] Yicong Liu, Yixiong Zou, Ruixuan Li, and Yuhua Li. Spectral decomposition and transformation for cross-domain few-shot learning. *Neural Networks*, 179:106536, 2024.

[30] Sharada P. Mohanty, David P. Hughes, and Marcel Salathé. Using deep learning for image-based plant disease detection. *Frontiers in Plant Science*, 7(September), September 2016. Publisher Copyright: © 2016 Mohanty, Hughes and Salathé.

[31] Mehrdad Noori, Milad Cheraghalikhani, Ali Bahri, Gustavo A. Vargas Hakim, David Osowiechi, Ismail Ben Ayed, and Christian Desrosiers. Tfs-vit: Token-level feature stylization for domain generalization. *Pattern Recognition*, 149:110213, 2024.

[32] Cheng Perng Phoo and Bharath Hariharan. Self-training for few-shot transfer across extreme task differences. In *International Conference on Learning Representations*, 2021.

[33] Jake Snell, Kevin Swersky, and Richard Zemel. Prototypical networks for few-shot learning. In *Proceedings of the International Conference on Neural Information Processing Systems*, pages 4080–4090, 2017.

[34] Nitish Srivastava, Geoffrey Hinton, Alex Krizhevsky, Ilya Sutskever, and Ruslan Salakhutdinov. Dropout: a simple way to prevent neural networks from overfitting. *The journal of machine learning research*, 15(1):1929–1958, 2014.

[35] Maryam Sultana, Muzammal Naseer, Muhammad Haris Khan, Salman Khan, and Fahad Shahbaz Khan. Self-distilled vision transformer for domain generalization. In *Proceedings of the Asian Conference on Computer Vision (ACCV)*, pages 3068–3085, December 2022.

[36] Yi Tay, Dara Bahri, Donald Metzler, Da-Cheng Juan, Zhe Zhao, and Che Zheng. Synthesizer: Rethinking self-attention for transformer models. In Marina Meila and Tong Zhang, editors, *Proceedings of the 38th International Conference on Machine Learning*, volume 139 of *Proceedings of Machine Learning Research*, pages 10183–10192. PMLR, 18–24 Jul 2021.

[37] Hugo Touvron, Matthieu Cord, Matthijs Douze, Francisco Massa, Alexandre Sablayrolles, and Herve Jegou. Training data-efficient image transformers &amp; distillation through attention. In *International Conference on Machine Learning*, volume 139, pages 10347–10357, July 2021.

[38] Hung-Yu Tseng, Hsin-Ying Lee, Jia-Bin Huang, and Ming-Hsuan Yang. Cross-domain few-shot classification via learned feature-wise transformation. In *Proceedings of the International Conference on Learning Representations*, 2020.

[39] Ashish Vaswani, Noam Shazeer, Niki Parmar, Jakob Uszkoreit, Llion Jones, Aidan N Gomez, Ł ukasz Kaiser, and Illia Polosukhin. Attention is all you need. In I. Guyon, U. Von Luxburg, S. Bengio, H. Wallach, R. Fergus, S. Vishwanathan, and R. Garnett, editors, *Advances in Neural Information Processing Systems*, volume 30. Curran Associates, Inc., 2017.

[40] Oriol Vinyals, Charles Blundell, Timothy Lillicrap, Koray Kavukcuoglu, and Daan Wierstra. Matching networks for one shot learning. In *Proceedings of the International Conference on Neural Information Processing Systems*, pages 3637–3645, 2016.

[41] Riccardo Volpi, Hongseok Namkoong, Ozan Sener, John C Duchi, Vittorio Murino, and Silvio Savarese. Generalizing to unseen domains via adversarial data augmentation. In S. Bengio, H. Wallach, H. Larochelle, K. Grauman, N. Cesa-Bianchi, and R. Garnett, editors, *Advances in Neural Information Processing Systems*, volume 31. Curran Associates, Inc., 2018.

[42] Reece Walsh, Islam Osman, and Mohamed S. Shehata. Masked embedding modeling with rapid domain adjustment for few-shot image classification. *IEEE Transactions on Image Processing*, 32:4907–4920, 2023.

[43] Haoqing Wang and Zhi-Hong Deng. Cross-domain few-shot classification via adversarial task augmentation. In Zhi-Hua Zhou, editor, *Proceedings of the Thirtieth International Joint Conference on Artificial Intelligence, IJCAI-21*, pages 1075–1081. International Joint Conferences on Artificial Intelligence Organization, 8 2021. Main Track.

[44] Xiaosong Wang, Yifan Peng, Le Lu, Zhiyong Lu, Mohammadhadi Bagheri, and Ronald M. Summers. Chestx-ray8: Hospital-scale chest x-ray database and benchmarks on weakly-supervised classification and localization of common thorax diseases. In *2017 IEEE Conference on Computer Vision and Pattern Recognition (CVPR)*. IEEE, July 2017.

[45] Kan Wu, Jinnian Zhang, Houwen Peng, Mengchen Liu, Bin Xiao, Jianlong Fu, and Lu Yuan. Tinyvit: Fast pretraining distillation for small vision transformers. In *European conference on computer vision (ECCV)*, 2022.

[46] Jinyu Yang, Jingjing Liu, Ning Xu, and Junzhou Huang. Tvt: Transferable vision transformer for unsupervised domain adaptation. In *Proceedings of the IEEE/CVF Winter Conference on Applications of Computer Vision (WACV)*, pages 520–530, January 2023.

[47] Li Yuan, Yunpeng Chen, Tao Wang, Weihao Yu, Yujun Shi, Zi-Hang Jiang, Francis E.H. Tay, Jiashi Feng, and Shuicheng Yan. Tokens-to-token vit: Training vision transformers from scratch on imagenet. In *Proceedings of the IEEE/CVF International Conference on Computer Vision (ICCV)*, pages 558–567, October 2021.

[48] Chongzhi Zhang, Mingyuan Zhang, Shanghang Zhang, Daisheng Jin, Qiang Zhou, Zhongang Cai, Haiyu Zhao, Xianglong Liu, and Ziwei Liu. Delving deep into the generalization of vision transformers under distribution shifts. In *2022 IEEE/CVF Conference on Computer Vision and Pattern Recognition (CVPR)*, pages 7267–7276, 2022.

[49] Zhenyu Zhang, Guangyao Chen, Yixiong Zou, Zhimeng Huang, Yuhua Li, and Ruixuan Li. Micm: Rethinking unsupervised pretraining for enhanced few-shot learning. In *Proceedings of the 32nd ACM International Conference on Multimedia*, MM '24, page 7686–7695, New York, NY, USA, 2024. Association for Computing Machinery.

[50] Fei Zhou, Peng Wang, Lei Zhang, Wei Wei, and Yanning Zhang. Revisiting prototypical network for cross domain few-shot learning. In *Proceedings of the IEEE/CVF Conference on Computer Vision and Pattern Recognition (CVPR)*, pages 20061–20070, June 2023.

[51] Jinghao Zhou, Chen Wei, Huiyu Wang, Wei Shen, Cihang Xie, Alan Yuille, and Tao Kong. Image BERT pre-training with online tokenizer. In *International Conference on Learning Representations*, 2022.

[52] Xiang Zhou, Yuan Zeng, and Yi Gong. Learning to scale temperature in masked self-attention for image inpainting. *arXiv preprint arXiv:2302.06130*, 2023.

[53] Yixiong Zou, Yicong Liu, Yiman Hu, Yuhua Li, and Ruixuan Li. Flatten long-range loss landscapes for cross-domain few-shot learning. In *Proceedings of the IEEE/CVF Conference on Computer Vision and Pattern Recognition (CVPR)*, pages 23575–23584, June 2024.

[54] Yixiong Zou, Shanghang Zhang, Yuhua Li, and Ruixuan Li. Margin-based few-shot class-incremental learning with class-level overfitting mitigation. In *Proceedings of the International Conference on Neural Information Processing Systems*, 2022.

# A Appendix / supplemental material

## A.1 Dataset Description

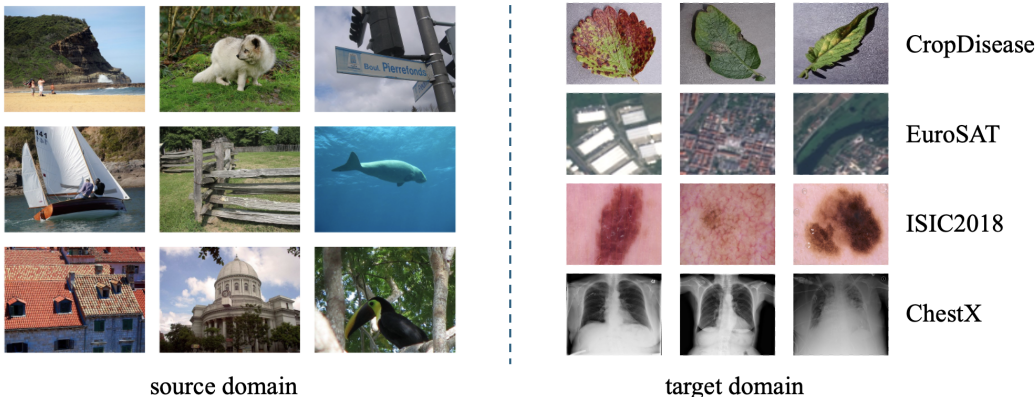

source domain                                         target domain

Figure 7: Samples of source domain *mini*Imagenet dataset (left) and target domain datasets (right), from top to bottom correspond to CropDiseases, EuroSAT, ISIC2018, and ChestX. We can see large domain gaps between source and target domains.

*mini***ImageNet** [40] is a subset derived from the larger ImageNet [6] dataset. It consists of 100 categories, each containing 600 natural images. Following the current works [2, 12], we employ the training set of *mini*ImageNet as the source domain dataset, consisting of 64 classes and 38,400 images. Additionally, as depicted in Fig. 7, we utilize datasets from four distinct domains as target domains following [12], including plant disease images, surface satellite imagery, skin disease images, and chest X-ray images. We will introduce them sequentially below.

**CropDiseases** [30] is a dataset for agricultural disease recognition, encompassing 38 distinct classes and a total of 43,456 images. The dataset comprises images of various crops, including infected and healthy plants, and corresponding disease category labels.

**EuroSAT** [13] is a comprehensive dataset comprising satellite imagery of the Earth. It encompasses a total of 27,000 images distributed across 10 distinct classes. The dataset offers a diverse range of geographical and topographical features.

**ISIC2018** [5] is a medical imaging dataset for skin lesion classification. The dataset consists of 10,015 images categorized into 7 distinct classes.

**ChestX** [44] is an X-ray medical imaging dataset for chest classification. The dataset consists of 25,847 images across 7 different classes.

## A.2 More experiments

### A.2.1 Applying Our Method to ViT Variants

Table 7: Our method with iBOT-pretrained ViT-S.

| Method | Shot | ChestX | ISIC2018 | EuroSAT | CropDiseases | Average |
|---|---|---|---|---|---|---|
| iBOT | 1 | 22.67 | 31.61 | 72.85 | 81.35 | 52.12 |
| **iBOT+Ours** | 1 | **23.01** | **34.69** | **73.04** | **82.39** | **53.28** |
| iBOT | 5 | 26.31 | 44.54 | 89.65 | 94.79 | 63.82 |
| **iBOT+Ours** | 5 | **27.63** | **51.06** | 89.21 | **95.20** | **65.78** |

We also apply our method to ViT-Small pretrained by iBOT[51] and ViT-Base pretrained by DINO[1]. iBOT is a self-supervised pre-training framework that learns semantic representations of images

Table 8: Our method with DINO-pretrained ViT-Base.

| Method | Shot | ChestX | ISIC2018 | EuroSAT | CropDiseases | Average |
|--------|------|--------|----------|---------|--------------|---------|
| DINO-B | 1 | 22.78 | 34.08 | 71.89 | 82.77 | 52.88 |
| **DINO-B+Ours** | 1 | **22.81** | **35.98** | **72.98** | **83.14** | **53.73** |
| DINO-B | 5 | 26.52 | 48.77 | 89.67 | 95.45 | 65.10 |
| **DINO-B+Ours** | 5 | **27.09** | **52.71** | **90.06** | **95.70** | **66.39** |

through Masked Image Modeling (MIM) and an online Tokenizer, enabling effective pre-training of vision Transformers without needing labeled data. The results are shown in Tab. 7. iBOT represents the iBOT-pretrained ViT baseline, while iBOT+Ours denotes our method applied to iBOT. As we can see, our method also shows considerable improvement on the ViT pre-trained with iBOT, with a 1.16 point increase in 1-shot and a 1.96 point increase in 5-shot. The results of our method applied to DINO-pretrained ViT-Base are shown in Tab. 8. DINO-B represents the DINO-pretrained ViT-Base baseline in our CDFSL task, and DINO-B+Ours denotes our method applied to DINO-pretrained ViT-Base. This also shows an improvement in ViT-Base with a 0.85 point increase in 1-shot and a 1.29 point increase in 5-shot.

### A.2.2 The Effectiveness of Our Attention

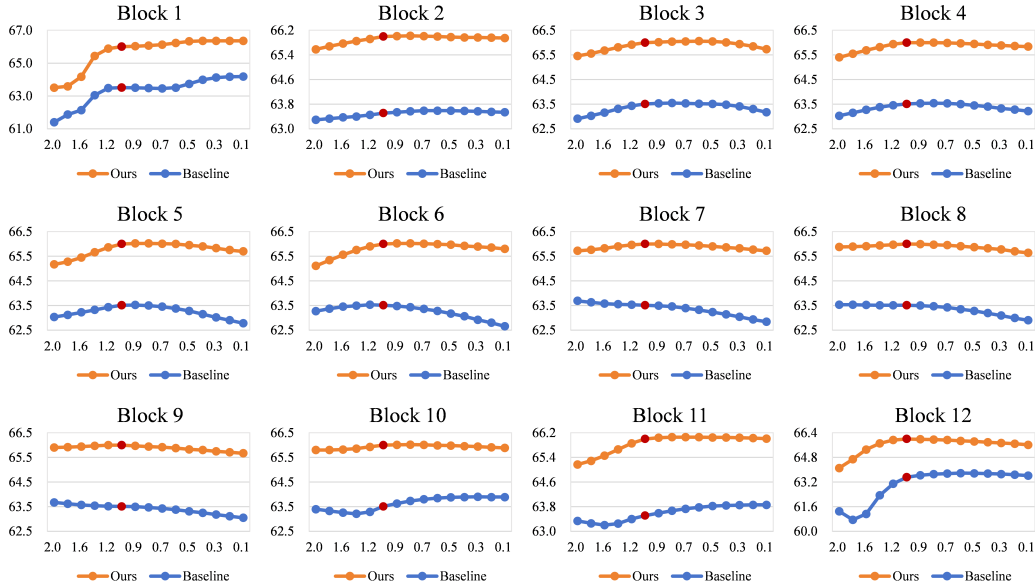

Figure 8: Average target domain accuracy vs. temperature. The red point represents the temperature is 1.0. If the attention is good enough, the attention adjustment will be trivial. Our method shows less reliance on the attention adjustments compared with the baseline method and achieves the best performance when the temperature is 1.0 in most blocks. This indicates the attention produced by our method is improved.

In Fig. 8, we illustrate the average target domain accuracy of the baseline method and our model with attention adjustment. The red point means the temperature is 1.0 (i.e., no temperature is applied). The accuracy of the baseline method increases with the temperature decrease from 2.0 to 0.0 in most blocks, which means temperature adjustment is important in the target domain. If the attention is already good enough, the impact of attention adjustment is trivial. As we can see, our method exhibits little variation in performance with temperature adjustment compared to the baseline method, and the highest accuracy is achieved when the temperature is 1.0 in most blocks. This verifies the effectiveness of our attention.

### A.3 Broader Impact

Our research introduces an improved ViT model based on attention temperature adjustment, aimed at addressing the ineffective target-domain attention caused by the query-key attention mechanism in CDFSL tasks. By suppressing the learning of query-key parameters and encouraging that of non-query-key parameters, our method significantly enhances the model's cross-domain transferability. This work is not only applicable to CDFSL but can also be extended to other domains, such as domain generalization, domain adaption, and few-shot class-incremental learning, where enhancing the transferability of self-attention is a prevalent challenge. While our method has been evaluated across four distinct target domains, providing a good initial assessment of our method's cross-domain applicability, the diversity of these domains may not encompass all possible real-world scenarios. Future work will aim to extend our evaluations to include a wider range of target domains to understand their performance in diverse real-world settings better.

